# Sparse Greedy
# Gaussian Process Regression

**Alex J. Smola*** 
RSISE and Department of Engineering
Australian National University
Canberra, ACT, 0200
*Alex.Smola@anu.edu.au*

**Peter Bartlett**
RSISE
Australian National University
Canberra, ACT, 0200
*Peter.Bartlett@anu.edu.au*

## Abstract

We present a simple sparse greedy technique to approximate the maximum a posteriori estimate of Gaussian Processes with much improved scaling behaviour in the sample size $m$. In particular, computational requirements are $O(n^2m)$, storage is $O(nm)$, the cost for prediction is $O(n)$ and the cost to compute confidence bounds is $O(nm)$, where $n \ll m$. We show how to compute a stopping criterion, give bounds on the approximation error, and show applications to large scale problems.

## 1 Introduction

Gaussian processes have become popular because they allow exact Bayesian analysis with simple matrix manipulations, yet provide good performance. They share with Support Vector machines and Regularization Networks the concept of regularization via Reproducing Kernel Hilbert spaces [3], that is, they allow the direct specification of the smoothness properties of the class of functions under consideration. However, Gaussian processes are not always the method of choice for large datasets, since they involve evaluations of the covariance function at $m$ points (where $m$ denotes the sample size) in order to carry out inference at a single additional point. This may be rather costly to implement — practitioners prefer to use only a small number of basis functions (i.e. covariance function evaluations).

Furthermore, the Maximum a Posteriori (MAP) estimate requires computation, storage, and inversion of the full $m \times m$ covariance matrix $K_{ij} = k(x_i, x_j)$ where $x_1, \ldots, x_m$ are training patterns. While there exist techniques [2, 8] to reduce the computational cost of finding an estimate to $O(km^2)$ rather than $O(m^3)$ when the covariance matrix contains a significant number of small eigenvalues, all these methods still require computation and storage of the full covariance matrix. None of these methods addresses the problem of speeding up the prediction stage (except for the rare case when the integral operator corresponding to the kernel can be diagonalized analytically [8]).

We devise a sparse greedy method, similar to those proposed in the context of wavelets [4], solutions of linear systems [5] or matrix approximation [7] that finds

*Supported by the DFG (Sm 62-1) and the Australian Research Council.

an approximation of the MAP estimate by expanding it in terms of a small subset of kernel functions $k(x_i, \cdot)$. Briefly, the technique works as follows: given a set of (already chosen) kernel functions, we seek the additional function that increases the posterior probability most. We add it to the set of basis functions and repeat until the maximum is approximated sufficiently well. A similar approach for a tight upper bound on the posterior probability gives a stopping criterion.

## 2 Gaussian Process Regression

Consider a finite set $X = \{x_1, \ldots x_m\}$ of inputs. In Gaussian Process Regression, we assume that for any such set there is a covariance matrix $K$ with elements $K_{ij} = k(x_i, x_j)$. We assume that for each input $x$ there is a corresponding output $y(x)$, and that these outputs are generated by

$$y(x) = t(x) + \xi \tag{1}$$

where $t(x)$ and $\xi$ are both normal random variables, with $\xi \sim \mathcal{N}(0, \sigma^2)$ and $\mathbf{t} = (t(x_1), \ldots, t(x_m))^\top \sim \mathcal{N}(0, K)$. We can use Bayes theorem to determine the distribution of the output $y(x)$ at a (new) input $x$. Conditioned on the data $(X, \mathbf{y})$, the output $y(x)$ is normally distributed. It follows that the mean of this distribution is the maximum a posteriori probability (MAP) estimate of $y$. We are interested in estimating this mean, and also the variance.

It is possible to give an equivalent parametric representation of $y$ that is more convenient for our purposes. We may assume that the vector $\mathbf{y} = (y(x_1), \ldots, y(x_m))^\top$ of outputs is generated by

$$\mathbf{y} = K\alpha + \xi, \tag{2}$$

where $\alpha \sim \mathcal{N}(0, K^{-1})$ and $\xi \sim \mathcal{N}(0, \sigma^2 \mathbf{1})$. Consequently the posterior probability $p(\alpha | \mathbf{y}, X)$ over the latent variables $\alpha$ is proportional to

$$\exp\left(-\tfrac{1}{2\sigma^2} \|\mathbf{y} - K\alpha\|^2\right) \exp\left(-\tfrac{1}{2}\alpha^\top K\alpha\right) \tag{3}$$

and the conditional expectation of $y(x)$ for a (new) location $x$ is $\mathbf{E}[y(x)|\mathbf{y}, X] = \mathbf{k}^\top \alpha_{\text{opt}}$, where $\mathbf{k}^\top$ denotes the vector $(k(x_1, x), \ldots, k(x_m, x))$ and $\alpha_{\text{opt}}$ is the value of $\alpha$ that maximizes (3). Thus, it suffices to compute $\alpha_{\text{opt}}$ before any predictions are required. The problem of choosing the MAP estimate of $\alpha$ is equivalent to the problem of minimizing the negative log-posterior,

$$\underset{\alpha \in \mathbb{R}^m}{\text{minimize}} \left[ -\mathbf{y}^\top K\alpha + \tfrac{1}{2}\alpha^\top \left(\sigma^2 K + K^\top K\right) \alpha \right] \tag{4}$$

(ignoring constant terms and rescaling by $\sigma^2$). It is easy to show that the mean of the conditional distribution of $y(x)$ is $\mathbf{k}^\top (K + \sigma^2 \mathbf{1})^{-1} \mathbf{y}$, and its variance is $k(x, x) + \sigma^2 - \mathbf{k}^\top (K + \sigma^2 \mathbf{1})^{-1} \mathbf{k}$ (see, for example, [2]).

## 3 Approximate Minimization of Quadratic Forms

For Gaussian process regression, searching for an approximate solution to (4) relies on the assumption that a set of variables whose posterior probability is close to that of the mode of the distribution will be a good approximation for the MAP estimate. The following theorem suggests a simple approach to estimating the accuracy of an approximate solution to (4). It uses an idea from [2] in a modified, slightly more general form.

**Theorem 1 (Approximation Bounds for Quadratic Forms)** *Denote by $K \in \mathbb{R}^{m \times m}$ a positive semidefinite matrix, $\mathbf{y}, \alpha \in \mathbb{R}^m$ and define the two quadratic forms*

$$Q(\alpha) := -\mathbf{y}^\top K\alpha + \frac{1}{2}\alpha^\top (\sigma^2 K + K^\top K)\alpha, \tag{5}$$

$$Q^*(\alpha) := -\mathbf{y}^\top \alpha + \frac{1}{2}\alpha^\top (\sigma^2 \mathbf{1} + K)\alpha. \tag{6}$$

*Suppose $Q$ and $Q^*$ have minima $Q_{\min}$ and $Q^*_{\min}$. Then for all $\alpha, \alpha^* \in \mathbb{R}^m$ we have*

$$Q(\alpha) \geq \quad Q_{\min} \quad \geq -\frac{1}{2}\|\mathbf{y}\|^2 - \sigma^2 Q^*(\alpha^*), \tag{7}$$

$$Q^*(\alpha^*) \geq \quad Q^*_{\min} \quad \geq \sigma^{-2}\left(-\frac{1}{2}\|\mathbf{y}\|^2 - Q(\alpha)\right), \tag{8}$$

*with equalities throughout when $Q(\alpha) = Q_{\min}$ and $Q^*(\alpha^*) = Q^*_{\min}$.*

Hence, by minimizing $Q^*$ in addition to $Q$ we can bound $Q$'s closeness to the optimum and vice versa.

**Proof** The minimum of $Q(\alpha)$ is obtained for $\alpha_{\mathrm{opt}} = (K + \sigma^2 \mathbf{1})^{-1}\mathbf{y}$ (which also minimizes $Q^*$), hence

$$Q_{\min} = -\frac{1}{2}\mathbf{y}^\top K(K + \sigma^2 \mathbf{1})^{-1}\mathbf{y} \text{ and } Q^*_{\min} = -\frac{1}{2}\mathbf{y}^\top (K + \sigma^2 \mathbf{1})^{-1}\mathbf{y}. \tag{9}$$

This allows us to combine $Q_{\min}$ and $Q^*_{\min}$ to $Q_{\min} + \sigma^2 Q^*_{\min} = -\frac{1}{2}\|\mathbf{y}\|^2$. Since by definition $Q(\alpha) \geq Q_{\min}$ for all $\alpha$ (and likewise $Q^*(\alpha^*) \geq Q^*_{\min}$ for all $\alpha^*$) we may solve $Q_{\min} + \sigma^2 Q^*_{\min}$ for either $Q$ or $Q^*$ to obtain lower bounds for each of the two quantities. This proves (7) and (8). ∎

Equation (7) is useful for computing an approximation to the MAP solution, whereas (8) can be used to obtain error bars on the estimate. To see this, note that in calculating the variance, the expensive quantity to compute is $-\mathbf{k}^\top (K + \sigma^2 \mathbf{1})^{-1}\mathbf{k}$. However, this can be found by solving

$$\underset{\alpha \in \mathbb{R}^m}{\text{minimize}} \left[ -\mathbf{k}^\top \alpha + \frac{1}{2}\alpha^\top \left( \sigma^2 \mathbf{1} + K \right) \alpha \right], \tag{10}$$

and the expression inside the parentheses is $Q^*(\alpha)$ with $\mathbf{y} = \mathbf{k}$ (see (6)). Hence, an approximate minimizer of (10) gives an *upper bound* on the error bars, and lower bounds can be obtained from (8).

In practice we will use the quantiy $\mathrm{gap}(\alpha, \alpha^*) := \frac{2(Q(\alpha) + \sigma^2 Q^*(\alpha^*) + \frac{1}{2}\|y\|^2)}{-Q(\alpha) + \sigma^2 Q^*(\alpha^*) + \frac{1}{2}\|y\|^2}$ , i.e. the relative size of the difference between upper and lower bound as stopping criterion.

## 4  A Sparse Greedy Algorithm

The central idea is that in order to obtain a faster algorithm, one has to reduce the number of free variables. Denote by $P \in \mathbb{R}^{m \times n}$ with $m \geq n$ and $m, n \in \mathbb{N}$ an extension matrix (i.e. $P^\top$ is a projection) with $P^\top P = \mathbf{1}$. We will make the ansatz

$$\alpha_P := P\beta \text{ where } \beta \in \mathbb{R}^n \tag{11}$$

and find solutions $\beta$ such that $Q(\alpha_P)$ (or $Q^*(\alpha_P)$) is minimized. The solution is

$$\beta_{\mathrm{opt}} = \left( P^\top \left( \sigma^2 K + K^\top K \right) P \right)^{-1} P^\top K^\top \mathbf{y}. \tag{12}$$

Clearly if $P$ is of rank $m$, this will also be the solution of (4) (the minimum negative log posterior for all $\alpha \in \mathbb{R}^m$). In all other cases, however, it is an approximation.

**Computational Cost of Greedy Decompositions**

For a given $P \in \mathbb{R}^{m \times n}$ let us analyze the computational cost involved in the estimation procedures. To compute (12) we need to evaluate $P^\top K \mathbf{y}$ which is $O(nm)$, $(KP)^\top (KP)$ which is $O(n^2 m)$ and invert an $n \times n$ matrix, which is $O(n^3)$. Hence the total cost is $O(n^2 m)$. Predictions then cost only $\mathbf{k}^\top \alpha$ which is $O(n)$. Using $P$ also to minimize $Q^*(P\beta^*)$ costs no more than $O(n^3)$, which is needed to upper-bound the log posterior.

For error bars, we have to approximately minimize (10) which can done for $\alpha = P\beta$ at $O(n^3)$ cost. If we compute $(PKP^\top)^{-1}$ beforehand, this can be done by at $O(n^2)$ and likewise for upper bounds. We have to minimize $-\mathbf{k}^\top KP\beta + \frac{1}{2}\beta^\top P^\top(\sigma^2 K + K^\top K)P\beta$ which costs $O(n^2 m)$ (once the inverse matrices have been computed, one may, however, use them to compute error bars at different locations, too, thus costing only $O(n^2)$). The lower bounds on the error bars may not be so crucial, since a bad estimate will only lead to overly conservative confidence intervals and not have any other negative effect. Finally note that all we ever have to compute and store is $KP$, i.e. the $m \times n$ submatrix of $K$ rather than $K$ itself. Table 1 summarizes the scaling behaviour of several optimization algorithms.

|  | Exact Solution | Conjugate Gradient [2] | Optimal Sparse Decomposition | Sparse Greedy Approximation |
|---|---|---|---|---|
| Memory | $O(m^2)$ | $O(m^2)$ | $O(nm)$ | $O(nm)$ |
| Initialization | $O(m^3)$ | $O(nm^2)$ | $O(n^2 m)$ | $O(\kappa n^2 m)$ |
| Pred. Mean | $O(m)$ | $O(m)$ | $O(n)$ | $O(n)$ |
| Error Bars | $O(m^2)$ | $O(nm^2)$ | $O(n^2 m)$ or $O(n^2)$ | $O(\kappa n^2 m)$ or $O(n^2)$ |

Table 1: Computational Cost of Optimization Methods. Note that $n \ll m$ and also note that the $n$ used in Conjugate Gradient, Sparse Decomposition, and Sparse Greedy Approximation methods will differ, with $n_{\mathrm{CG}} \le n_{\mathrm{SD}} \le n_{\mathrm{SGA}}$ since the search spaces are more restricted. $\kappa = 60$ gives near-optimal results.

**Sparse Greedy Approximation**

Several choices for $P$ are possible, including choosing the principal components of $K$ [8], using conjugate gradient descent to minimize $Q$ [2], symmetric Cholesky factorization [1], or using a sparse greedy approximation of $K$ [7]. Yet these methods have the disadvantage that they either do not take the specific form of $\mathbf{y}$ into account [8, 7] or lead to expansions that cost $O(m)$ for prediction and require computation *and storage* of the full matrix [8, 2].

If we require a sparse expansion of $y(x)$ in terms of $k(x_i, x)$ (i.e. many $\alpha_i$ in $y = \mathbf{k}^\top \alpha$ will be 0) we must consider matrices $P$ that are a collection of unit vectors $\mathbf{e}_i$ (here $(\mathbf{e}_i)_j = \delta_{ij}$). We use a greedy approach to find a good approximation. First, for $n = 1$, we choose $P = e_i$ such that $Q(P\beta)$ is minimal. In this case we could permit ourselves to consider *all possible* indices $i \in \{1, \ldots m\}$ and find the best one by trying out all of them. Next assume that we have found a good solution $P\beta$ where $P$ contains $n$ columns. In order to improve this solution, we may expand $P$ into the matrix $P_{\mathrm{new}} := [P_{\mathrm{old}}, \mathbf{e}_i] \in \mathbb{R}^{m \times (n+1)}$ and seek the best $\mathbf{e}_i$ such that $P_{\mathrm{new}}$ minimizes $\min_\beta Q(P_{\mathrm{new}}\beta)$. (Performing a full search over all possible $n + 1$ out of $m$ indices would be too costly.) This greedy approach to finding a sparse approximate solution is described in Algorithm 1. The algorithm also maintains an approximate minimum of $Q^*$, and exploits the bounds of Theorem 1 to determine when the approximation is sufficiently accurate. (Note that we leave unspecified how the subsets $M \subseteq I, M^* \subseteq I^*$ are chosen. Assume for now that we choose $M = I, M^* = I^*$, the full set of indices that have not yet been selected.) This method is very similar to Matching Pursuit [4] or iterative reduced set Support Vector algorithms [6], with the difference that the target to be approximated (the full solution $\alpha$) is only given implicitly via $Q(\alpha)$.

**Approximation Quality**

Natarajan [5] studies the following Sparse Linear Approximation problem: Given $A \in \mathbb{R}^{m \times n}$, $b \in \mathbb{R}^m$, $\epsilon > 0$, find $x \in \mathbb{R}^n$ with minimal number of nonzero entries such that $\|Ax - b\|_2 \le \epsilon$.

If we define $A := (\sigma^2 K + K^\top K)^{\frac{1}{2}}$ and $b := A^{-1} K\mathbf{y}$, then we may write $Q(\alpha) = \frac{1}{2}\|b - A\alpha\|^2 + c$ where $c$ is a constant independent of $\alpha$. Thus the problem of sparse approximate minimization of $Q(\alpha)$ is a special case of Natarajan's problem (where the matrix $A$ is square, symmetric, and positive definite). In addition, the algorithm considered by Natarajan in [5] involves sequentially choosing columns of $A$ to maximally decrease $\|Ax - b\|$. This is clearly equivalent to the sparse greedy algorithm described above. Hence, it is straightforward to obtain the following result from Theorem 2 in [5].

**Theorem 2 (Approximation Rate)** *Algorithm 1 achieves* $Q(\alpha) \leq Q(\alpha_{\mathrm{opt}}) + \epsilon$ *when $\alpha$ has*

$$n \leq \frac{18 n^*(\epsilon/4)}{\lambda_1^2} \ln\left(\frac{\|A^{-1} K\mathbf{y}\|}{\epsilon}\right)$$

*non-zero components, where $n^*(\epsilon/4)$ is the minimal number of nonzero components in vectors $\alpha$ for which $Q(\alpha) \leq Q(\alpha_{\mathrm{opt}}) + \epsilon/4$, $A = (\sigma^2 K + K^\top K)^{1/2}$, and $\lambda_1$ is the minimum of the magnitudes of the singular values of $\mathbf{A}$, the matrix obtained by normalizing the columns of $A$.*

### Randomized Algorithms for Subset Selection

Unfortunately, the approximation algorithm considered above is still too expensive for large $m$ since each search operation involves $\Omega(m)$ indices. Yet, if we are satisfied with finding a *relatively good* index rather than the *best*, we may resort to selecting a random subset of size $\kappa \ll m$. In Algorithm 1, this corresponds to choosing $M \subseteq I, M^* \subseteq I^*$ as random subsets of size $\kappa$. In fact, a constant value of $\kappa$ will typically suffice. To see why, we recall a simple lemma from [7]: the cumulative distribution function of the maximum of $m$ i.i.d. random variables $\xi_1, \ldots, \xi_m$ is $F(\cdot)^m$, where $F(\cdot)$ is the cdf of $\xi_i$. Thus, in order to find a column to add to $P$ that is with probability 0.95 among the best 0.05 of all such columns, a random subsample of size $\lceil \log 0.05 / \log 0.95 \rceil = 59$ will suffice.

---

**Algorithm 1** Sparse Greedy Quadratic Minimization.

---

**Require:** Training data $X = \{x_1, \ldots, x_m\}$, Targets $\mathbf{y}$, Noise $\sigma^2$, Precision $\epsilon$
    Initialize index sets $I, I^* = \{1, \ldots, m\}$; $S, S^* = \emptyset$.
    **repeat**
        Choose $M \subseteq I, M^* \subseteq I^*$.
        Find $\arg\min_{i \in M} Q\left([P, e_i]\beta_{\mathrm{opt}}\right), \arg\min_{i^* \in M^*} Q^*\left([P^*, e_{i^*}]\beta_{\mathrm{opt}}^*\right)$.
        Move $i$ from $I$ to $S$, $i^*$ from $I^*$ to $S^*$.
        Set $P := [P, e_i]$, $P^* := [P^*, e_{i^*}]$.
    **until** $Q(P\beta_{\mathrm{opt}}) + \sigma^2 Q^*(P\beta_{\mathrm{opt}}^*) + \frac{1}{2}\|\mathbf{y}\|^2 \leq \frac{\epsilon}{2}(|Q(P\beta_{\mathrm{opt}})| + |\sigma^2 Q^*(P\beta_{\mathrm{opt}}^*) + \frac{1}{2}\|\mathbf{y}\|^2|$
**Output:** Set of indices $S$, $\beta_{\mathrm{opt}}$, $(P^\top K P)^{-1}$, and $(P^\top (K^\top K + \sigma^2 K) P)^{-1}$.

---

### Numerical Considerations

The crucial part is to obtain the values of $Q(P\beta_{\mathrm{opt}})$ cheaply (with $P = [P_{\mathrm{old}}, \mathbf{e}_i]$), provided we solved the problem for $P_{\mathrm{old}}$. From (12) one can see that all that needs to be done is a rank-1 update on the inverse. In the following we will show that this can be obtained in $O(mn)$ operations, provided the inverse of the smaller subsystem is known. Expressing the relevant terms using $P_{\mathrm{old}}$ and $\mathbf{k}_i$ we obtain

$$P^\top K^\top \mathbf{y} = [P_{\mathrm{old}}, \mathbf{e}_i]^\top K^\top \mathbf{y} = (P_{\mathrm{old}}^\top K^\top \mathbf{y}, \mathbf{k}_i^\top \mathbf{y})$$

$$P^\top \left(K^\top K + \sigma^2 K\right) P = \begin{bmatrix} P_{\mathrm{old}}^\top \left(K^\top K + \sigma^2 K\right) P_{\mathrm{old}} & P_{\mathrm{old}}^\top \left(K^\top + \sigma^2 \mathbf{1}\right) \mathbf{k}_i \\ \mathbf{k}_i^\top (K + \sigma^2 \mathbf{1}) P_{\mathrm{old}} & \mathbf{k}_i^\top \mathbf{k}_i + \sigma^2 K_{ii} \end{bmatrix}$$

Thus computation of the terms costs only $O(nm)$, given the values for $P_{\text{old}}$. Furthermore, it is easy to verify that we can write the inverse of a symmetric positive semidefinite matrix as

$$\begin{bmatrix} A & B \\ B^\top & C \end{bmatrix}^{-1} = \begin{bmatrix} A^{-1} + (A^{-1}B)^\top \gamma (A^{-1}B) & -\gamma(A^{-1}B) \\ -(\gamma(A^{-1}B))^\top & \gamma \end{bmatrix} \quad (13)$$

where $\gamma := (C + B^\top A^{-1}B)^{-1}$. Hence, inversion of $P^\top \left(K^\top K + \sigma^2 K\right) P$ costs only $O(n^2)$. Thus, to find $P$ of size $m \times n$ takes $O(\kappa n^2 m)$ time. For the error bars, $(P^\top KP)^{-1}$ will generally be a good starting value for the minimization of (10), so the typical cost for (10) will be $O(\tau mn)$ for some $\tau < n$, rather than $O(mn^2)$. Finally, for added numerical stability one may want to use an incremental Cholesky factorization in (13) instead of the inverse of a matrix.

## 5   Experiments and Discussion

We used the `Abalone` dataset from the UCI Repository to investigate the properties of the algorithm. The dataset is of size 4177, split into 4000 training and 177 testing split to analyze the numerical performance, and a $(3000, 1177)$ split to assess the generalization error (the latter was needed in order to be able to invert (and keep in memory) the full matrix $K + \sigma^2 \mathbf{1}$ for a comparison). The data was rescaled to zero mean and unit variance coordinate-wise. Finally, the gender encoding in `Abalone` (male/female/infant) was mapped into $\{(1,0,0), (0,1,0), (0,0,1)\}$.

In all our experiments we used Gaussian kernels $k(x, x') = \exp(-\frac{\|x-x'\|^2}{2\omega^2})$ as covariance kernels. Figure 1 analyzes the speed of convergence for different $\kappa$.

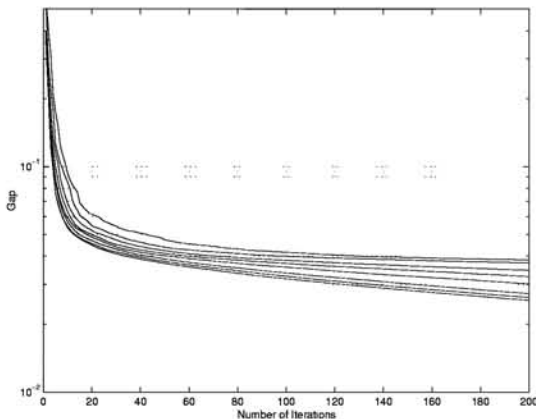

Figure 1: Speed of Convergence. We plot the size of the gap between upper and lower bound of the log posterior $(\text{gap}(\alpha, \alpha^*))$ for the first 4000 samples from the Abalone dataset $(\sigma^2 = 0.1$ and $2\omega^2 = 10)$. From top to bottom: subsets of size 1, 2, 5, 10, 20, 50, 100, 200. The results were averaged over 10 runs. The relative variance of the gap size was less than 10%.

One can see that that subsets of size 50 and above ensure rapid convergence.

For the optimal parameters $(2\sigma^2 = 0.1$ and $2\omega^2 = 10$, chosen after [7]) the average test error of the sparse greedy approximation trained until $\text{gap}(\alpha, \alpha^*) < 0.025$ on a $(3000, 1177)$ split (the results were averaged over ten independent choices of training sets.) was $1.785 \pm 0.32$, slightly worse than for the GP estimate $(1.782 \pm 0.33)$. The log posterior was $-1.572 \cdot 10^5 (1 \pm 0.005)$, the optimal value $-1.571 \cdot 10^5 (1 \pm 0.005)$. Hence for all practical purposes full inversion of the covariance matrix and the sparse greedy approximation have statistically indistinguishable generalization performance.

In a third experiment (Table 2) we analyzed the number of basis functions needed to minimize the log posterior to $\text{gap}(\alpha, \alpha^*) < 0.025$, depending on different choices of the kernel width $\sigma$. In all cases, less than 10% of the kernel functions suffice to

find a good minimizer of the log posterior, for the error bars, even less than 2% are sufficient. This is a dramatic improvement over previous techniques.

| Kernel width $2\omega^2$ | 1 | 2 | 5 | 10 | 20 | 50 |
|---|---|---|---|---|---|---|
| Kernels for log-posterior | 373 | 287 | 255 | 257 | 251 | 270 |
| Kernels for error bars | 79±61 | 49±43 | 26±27 | 17±16 | 12±9 | 8±5 |

Table 2: Number of basis functions needed to minimize the log posterior on the Abalone dataset (4000 training samples), depending on the width of the kernel $\omega$. Also, number of basis functions required to approximate $\mathbf{k}^\top (K + \sigma^2 \mathbf{1})^{-1}\mathbf{k}$ which is needed to compute the error bars. We averaged over the remaining 177 test samples.

To ensure that our results were not dataset specific and that the algorithm scales well we tested it on a larger synthetic dataset of size 10000 in 20 dimensions distributed according to $\mathcal{N}(0,1)$. The data was generated by adding normal noise with variance $\sigma^2 = 0.1$ to a function consisting of 200 randomly chosen Gaussians of width $2\omega^2 = 40$ and normally distributed coefficients and centers.

We purposely chose an *inadequate* Gaussian process prior (but correct noise level) of Gaussians with width $2\omega^2 = 10$ in order to avoid trivial sparse expansions. After 500 iterations (i.e. after using 5% of all basis functions) the size of the gap$(\alpha, \alpha^*)$ was less than 0.023 (note that this problem is too large to be solved exactly).

We believe that sparse greedy approximation methods are a key technique to scale up Gaussian Process regression to sample sizes of 10.000 and beyond. The techniques presented in the paper, however, are by no means limited to regression. Work on the solutions of dense quadratic programs and classification problems is in progress. The authors thank Bob Williamson and Bernhard Schölkopf.

# References

[1] S. Fine and K. Scheinberg. Efficient SVM training using low-rank kernel representation. Technical report, IBM Watson Research Center, New York, 2000.

[2] M. Gibbs and D.J.C. Mackay. Efficient implementation of gaussian processes. Technical report, Cavendish Laboratory, Cambridge, UK, 1997.

[3] F. Girosi. An equivalence between sparse approximation and support vector machines. *Neural Computation*, 10(6):1455–1480, 1998.

[4] S. Mallat and Z. Zhang. Matching Pursuit in a time-frequency dictionary. *IEEE Transactions on Signal Processing*, 41:3397–3415, 1993.

[5] B. K. Natarajan. Sparse approximate solutions to linear systems. *SIAM Journal of Computing*, 25(2):227–234, 1995.

[6] B. Schölkopf, S. Mika, C. Burges, P. Knirsch, K.-R. Müller, G. Rätsch, and A. Smola. Input space vs. feature space in kernel-based methods. *IEEE Transactions on Neural Networks*, 10(5):1000 – 1017, 1999.

[7] A.J. Smola and B. Schölkopf. Sparse greedy matrix approximation for machine learning. In P. Langley, editor, *Proceedings of the 17th International Conference on Machine Learning*, pages 911 – 918, San Francisco, 2000. Morgan Kaufman.

[8] C.K.I. Williams and M. Seeger. The effect of the input density distribution on kernel-based classifiers. In P. Langley, editor, *Proceedings of the Seventeenth International Conference on Machine Learning*, pages 1159 – 1166, San Francisco, California, 2000. Morgan Kaufmann.
